# Categorization Under Complexity: A Unified MDL Account of Human Learning of Regular and Irregular Categories

**David Fass**
Department of Psychology
Center for Cognitive Science
Rutgers University
Piscataway, NJ 08854
dfass@ruccs.rutgers.edu

**Jacob Feldman***
Department of Psychology
Center for Cognitive Science
Rutgers University
Piscataway, NJ 08854
jacob@ruccs.rutgers.edu

## Abstract

We present an account of human concept learning—that is, learning of categories from examples—based on the principle of minimum description length (MDL). In support of this theory, we tested a wide range of two-dimensional concept types, including both regular (simple) and highly irregular (complex) structures, and found the MDL theory to give a good account of subjects' performance. This suggests that the *intrinsic complexity* of a concept (that is, its description length) systematically influences its learnability.

## 1 The Structure of Categories

A number of different principles have been advanced to explain the manner in which humans learn to categorize objects. It has been variously suggested that the underlying principle might be the *similarity structure* of objects [1], the manipulability of *decision boundaries* [2], or Bayesian inference [3][4]. While many of these theories are mathematically well-grounded and have been successful in explaining a range of experimental findings, they have commonly only been tested on a narrow collection of concept types similar to the simple unimodal categories of Figure 1(a–c).

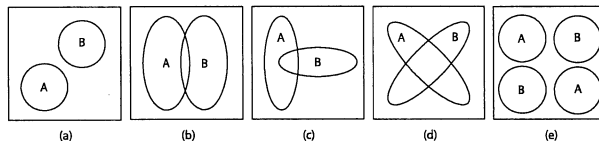

Figure 1: Categories similar to those previously studied. Lines represent contours of equal probability. All except (e) are unimodal.

*http://ruccs.rutgers.edu/~jacob/feldman.html

Moreover, in the scarce research that has ventured to look beyond simple category types, the goal has largely been to investigate categorization performance for *isolated* irregular distributions, rather than to present a survey of performance across a *range* of interesting distributions. For example, Nosofsky has previously examined the "criss-cross" category of Figure 1(d) and a diagonal category similar to Concept 3 of Figure 2, as well as some other multimodal categories [5][6]. While these individual category structures are no doubt theoretically important, they in no way exhaust the range of possible concept structures. Indeed, if we view $n$-dimensional Cartesian space as the canvas upon which a category may be represented, then any set of manifolds in that space may be considered as a potential category [7]. It is therefore natural to ask whether one such category-manifold is *in principle* easier or more difficult to learn than another. Since previous investigations have never considered any reasonably broad range of category structures, they have never been in a position to answer this question.

In this paper we present a theory for human categorization, based on the MDL principle, that is much better equipped to answer questions about the intrinsic learnability of both structurally regular and structurally irregular categories. In support of this theory we briefly present an experiment testing human subjects' learning of a range of concept types defined over a continuous two-dimensional feature space, including both highly regular and highly irregular structures. We find that our MDL-based theory gives a good account of human learning for these concepts, and that descriptive complexity accurately predicts the subjective difficulty of the various concept types tested.

## 2  Previous Investigations of Category Structure

The role of category structure in determining learnability has not been overlooked entirely in the literature; in fact, the intrinsic structure of *binary-featured* categories has been investigated quite thoroughly. The classic work by Shepard et al. [8] showed that human performance in learning such Boolean categories varies greatly depending on the intrinsic logical structure of the concept. More recently, we have shown that this performance is well-predicted by the intrinsic *Boolean complexity* of each concept, given by the length of the shortest Boolean formula that describes the objects in the category [9]. This result suggests that a principle of simplicity or parsimony, manifested as a minimization of complexity, might play an important role in human category learning.

The details of Boolean complexity analysis do not generalize easily to the type of continuous feature spaces we wish to investigate here. Thus a new approach is required, similar in general spirit but differing in the mathematics. Our goals are therefore (1) to deploy a complexity minimization technique such as MDL to quantify the complexity of categories defined over continuous features, and (2) to investigate the influence of complexity on human category learning by testing a range of concept types differing widely in intrinsic complexity.

## 3  Experiment

While the MDL principle that we plan to employ is applicable to concepts of any dimension, for reasons of convenience this experiment is limited to category structures that can be formed within a two-dimensional feature space. This feature space is discretized into a $4 \times 4$ grid from which a legitimate category can be specified by the selection of any four grid squares. Our motivation for discretizing the feature space is to place a constraint on possible category structure that will facilitate the computation of a complexity measure; this does not restrict the range of possible *feature* values that can be adopted by stimuli. In principle, feature values are limited only by machine precision, but as a matter of convenience

we restrict features to adopting one of 1000 possible values in the range [0,1].

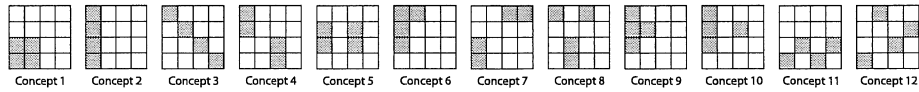

Figure 2: Abstract concepts used in experiment.

The particular 12 abstract category structures ("concepts") examined in the experiment are shown in Figure 2. These concepts were considered to be individually interesting (from a cross-theoretical perspective) and jointly representative of the broader range of available concepts. The two categories in each concept are referred to as "positive" and "negative." The positive category is represented by the dark-shaded regions, and the corresponding negative category is its complement. Note that in many cases the categories are "disconnected" or multimodal. Nevertheless, these categories are not in any sense "probabilistic" or "ill-defined"; a given point in feature space is *always* either positive or negative.

During the experiment, each stimulus is drawn randomly from the feature space and is labeled "positive" or "negative" based on the region from which it was drawn. Uniform sampling is used, so all 12 categories of Figure 2 have the same base rate for positives, $P(\text{positive}) = \frac{4}{16} = \frac{1}{4}$.

The experiment itself was clothed as a video game that required subjects to discriminate between two classes of spaceships, "Ally" and "Enemy," by destroying Enemy ships and quick-landing Allied ships. Each subject (14 total) played 12 five-minute games in which the distribution of Allies and Enemies corresponded (in random order) to the 12 concepts of Figure 2. The physical features of the spaceships in all cases were the height of the "tube" and the radius of the "pod." As shown in Figure 3, these physical features are mapped randomly onto the abstract feature space such that the experimental concepts may be any rigid rotation or reflection of the abstract concepts in Figure 2.

# 4   Derivation of the MDL Principle

The MDL principle is largely due to Rissanen [10] and is easily shown to be a consequence of optimal Bayesian inference [11]. While several Bayesian algorithms have previously been proposed as models of human concept learning [3][4], the implications of the MDL principle for human learning have only recently come under scrutiny [12][13]. We briefly review the relevant theory.

According to Bayes rule, a learner ought to select the category hypothesis $H$ that *maximizes*

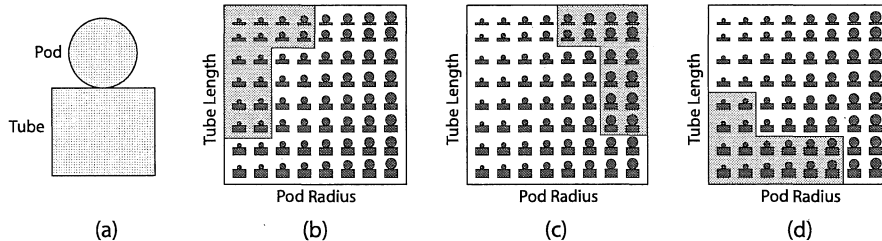

Figure 3: (a) A spaceship. (b–d) Three possible instantiations of Concept 6 from Figure 2.

the posterior $P(H \mid D)$, where $D$ is the data, and

$$P(H \mid D) = \frac{P(D \mid H)P(H)}{P(D)} \tag{1}$$

Taking negative logarithms of both sides, we obtain

$$-\log P(H \mid D) = -\log P(D \mid H) - \log P(H) + \log P(D) \tag{2}$$

The problem of maximizing $P(H \mid D)$ is thus identical to the problem of *minimizing* $-\log P(H \mid D)$. Since $\log P(D)$ is constant for all hypotheses, its value does not enter into the minimization problem, and we can state that the hypothesis of choice ought to be such as to minimize the quantity

$$-\log P(D \mid H) - \log P(H) \tag{3}$$

If we follow Rissanen and regard the quantity $-\log P(x)$ as the description length of $x$, $DL(x)$, then Equation 3 instructs us to select the hypothesis that minimizes the total description length

$$DL(D \mid H) + DL(H) \tag{4}$$

What this means is that the hypothesis that is optimal from the standpoint of the Bayesian decision maker is the same hypothesis that yields the most compact two-part code in Equation 4. Thus, besides the merits of brevity for its own sake, we see that maximal descriptive compactness also corresponds to maximal inferential power. It is this equivalence between description length and inference that leads us to investigate the role of descriptive complexity in the domain of concept learning.

## 5   Theory

In order to investigate the complexity of the 12 concepts of Figure 2, Equation 4 indicates that we need to analyze (1) the description length of a hypothesis for each concept, $DL(H)$, and (2) the description length of the concept given the hypothesis, $DL(D \mid H)$. We discuss these in sequence.

### 5.1   The Hypothesis Description Length, $DL(H)$

In order to compute $DL(H)$, we first fix a language[1] within which hypotheses about the category structure can be expressed. We choose to use the "rectangle language" whose alphabet (Table 1) consists of 10 *classes* of symbols representing the 10 different sizes of rectangle that can be composited within a 4×4 grid: 1×1, 1×2, 1×3, 1×4, 2×2, 2×3, 2×4, 3×3, 3×4, and 4×4.[2] Each member of the class "$m \times n$" is an $m \times n$ or $n \times m$ rectangle situated at a particular position in the 4×4 grid. We allow a given hypothesis to be represented by up to four distinct rectangles (i.e., four symbols).

Having specified a language, the issue is now the *length* of the hypothesis code. The derivation above suggests that a codelength of $-\log P(x)$ be assigned to each symbol $x$, which corresponds to the so-called Shannon code. We therefore proceed to compute the Shannon codelengths for the rectangle alphabet of Table 1.[3]

Table 1: Rectangle alphabet. The third and fourth columns show the probability that the source generates a given member of the class "$m \times n$" and the corresponding codelength.

| Rectangle Class | Possible Locations | Probability | Codelength |
|:---:|:---:|:---:|:---:|
| $1 \times 1$ | 16 | $\frac{1}{10} \cdot \frac{1}{16}$ | $-\log\left(\frac{1}{160}\right)$ |
| $1 \times 2$ | 24 | $\frac{1}{10} \cdot \frac{1}{24}$ | $-\log\left(\frac{1}{240}\right)$ |
| $1 \times 3$ | 16 | $\frac{1}{10} \cdot \frac{1}{16}$ | $-\log\left(\frac{1}{160}\right)$ |
| $1 \times 4$ | 8 | $\frac{1}{10} \cdot \frac{1}{8}$ | $-\log\left(\frac{1}{80}\right)$ |
| $2 \times 2$ | 9 | $\frac{1}{10} \cdot \frac{1}{9}$ | $-\log\left(\frac{1}{90}\right)$ |
| $2 \times 3$ | 12 | $\frac{1}{10} \cdot \frac{1}{12}$ | $-\log\left(\frac{1}{120}\right)$ |
| $2 \times 4$ | 6 | $\frac{1}{10} \cdot \frac{1}{6}$ | $-\log\left(\frac{1}{60}\right)$ |
| $3 \times 3$ | 4 | $\frac{1}{10} \cdot \frac{1}{4}$ | $-\log\left(\frac{1}{40}\right)$ |
| $3 \times 4$ | 4 | $\frac{1}{10} \cdot \frac{1}{4}$ | $-\log\left(\frac{1}{40}\right)$ |
| $4 \times 4$ | 1 | $\frac{1}{10} \cdot 1$ | $-\log\left(\frac{1}{10}\right)$ |

Computing these codelengths requires that we specify the probability mass function of a source, $P(x)$. It is convenient for this purpose (and compatible with the subject's perspective) to imagine that the concepts in Figure 2 are produced by a "concept generator," an information source whose parameters are essentially unknown. A reasonable assumption is that the source randomly selects a rectangle *class* with uniform probability, and then selects an individual member of the chosen class also with uniform probability. Since there are 10 classes, the assumption regarding class selection places a prior on each rectangle class of $P(m \times n) = \frac{1}{10}$.

Moreover, the assumption of uniform within-class sampling means that in order to encode any individual rectangle, we need only consider the cardinality of the class to which it belongs. We now recall that the individual rectangles of the class "$m \times n$" differ only in their positions within the 4×4 grid. Therefore, the cardinality of the class "$m \times n$" is equal to the number of unique ways $N_{m \times n}$ in which an $m \times n$ or $n \times m$ rectangle can be selected from a 4×4 grid, where

$$N_{m \times n} = \begin{cases} (5-m)(5-n), & m = n \\ 2(5-m)(5-n), & m \neq n \end{cases} \qquad (5)$$

Thus, the probability associated with an individual rectangle of class "$m \times n$" is $\frac{P(m \times n)}{N_{m \times n}}$. The corresponding Shannon codelengths are shown next to these probabilities in Table 1. The description length of a particular hypothesis is the summed codeword lengths for all the rectangles (up to four) that are comprised by the hypothesis.

## 5.2  The Likelihood Description Length, $DL(D \mid H)$

The second part of the two-part MDL code is the description of the concept with respect to the selected hypothesis, corresponding to the Bayes *likelihood*. There are several possible approaches to computing $DL(D \mid H)$; we discuss one that is particularly straightforward.

We recall that a hypothesis $H$ is composed of up to four rectangular regions. Computing $DL(D \mid H)$ therefore involves describing that portion of the positive category that falls within each rectangular hypothesis region. This is conceptually the same problem that we faced in computing $DL(H)$ above, except that the region of interest for $DL(H)$ was fixed

Table 2: Minimum description lengths for the 12 abstract concepts.

| Concept | MDL Codelength | MDL Hypothesis | MDL Concept |
|---------|---------------|----------------|-------------|
| 1 | 8.0768 bits |  |  |
| 2 | 8.3219 bits | 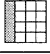 | 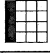 |
| 3 | 27.3236 bits | 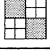 | 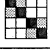 |
| 4 | 17.8138 bits | 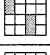 | 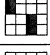 |
| 5 | 16.5216 bits | 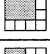 |  |
| 6 | 14.4919 bits | 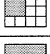 | 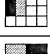 |
| 7 | 17.1357 bits | 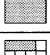 | 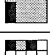 |
| 8 | 22.5687 bits |  |  |
| 9 | 14.4919 bits |  | 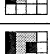 |
| 10 | 15.0768 bits | 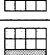 | 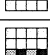 |
| 11 | 27.1946 bits | 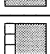 |  |
| 12 | 28.1536 bits | 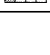 | 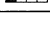 |

at 4×4, while the regions for $DL(D \mid H)$ can be of any dimension 4×4 and smaller. Guided by this analogy, we follow the procedure of the previous section to compute an appropriate probability mass function. Since $DL(D \mid H)$ must capture *just* the positive squares in the hypothesis region (a maximum of four squares), the only rectangle classes needed in the alphabet are those of size four: 1×1, 1×2, 1×3, 1×4, and 2×2.

# 6 Minimum Description Lengths for Experimental Concepts

Applying the MDL analysis above to the concepts in Figure 2 requires that we compute the total description length $DL(D \mid H) + DL(H)$ corresponding to all viable hypotheses for each concept. The hypothesis $H$ corresponding to the shortest total codelength $DL(D \mid H) + DL(H)$ for each concept is the *MDL hypothesis*.[4] The MDL hypotheses for all 12 concepts are shown in Table 2 along with the corresponding minimum codelengths. It can be observed that while for some concepts the MDL hypothesis precisely conforms to the true positive category (meaning that almost all of the concept information is carried in the hypothesis code), for the majority of concepts the MDL hypothesis is *broader* than the true category region (meaning that the concept information is distributed between the hypothesis and likelihood codes).

# 7 Results

For each game played by the subject (i.e., each concept in Figure 2), an overall measure of performance ($d'$) is computed.[5] Figure 4 shows performance for all subjects and all concepts as a function of the concept complexities (MDL codelengths) in Table 2. There is an evident decrease in performance with increasing complexity, which a regression analysis shows to be highly significant ($R^2 = .384$, $F(1, 166) = 103.375$, $p < .000001$), meaning that the linear trend in the plot is very unlikely to be a statistical accident. Thus, the MDL complexity predicts the subjective difficulty of learning across a broad range of concepts.

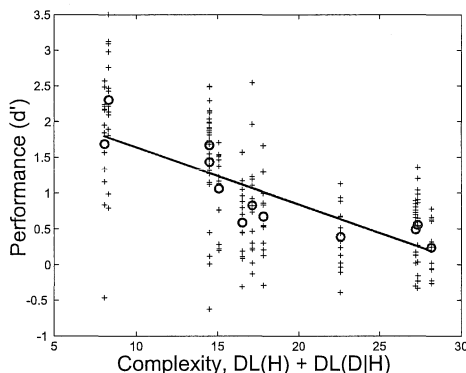

Figure 4: Performance vs. complexity for all 14 subjects. The $d'$ performance for each concept is indicated by a '+' and the mean $d'$ for each concept is indicated by an 'O'.

We mention that the MDL approach described here can be further modified to make "real-time" predictions of how subjects will categorize each new stimulus. In the most simplistic approach, the prediction for each new stimulus $x$ is made based on the MDL hypothesis prevailing at the time that stimulus is observed. Correlation between this MDL prediction and the subject's actual decision is found to be highly significant ($p \leq .002$) for each of the 12 concept types. The Pearson $r$ statistics are given below:

| Concept #: | 1 | 2 | 3 | 4 | 5 | 6 | 7 | 8 | 9 | 10 | 11 | 12 |
|---|---|---|---|---|---|---|---|---|---|---|---|---|
| **Pearson $r$:** | .46 | .47 | .19 | .18 | .20 | .51 | .18 | .14 | .34 | .32 | .32 | .05 |

Figure 5 illustrates the behavior of the real-time MDL algorithm. Simulations for a variety of data sets can be found at `http://ruccs.rutgers.edu/~dfass/mdlmovies.html`.

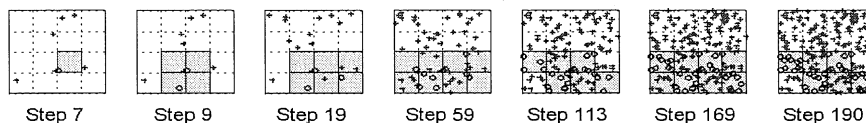

Step 7    Step 9    Step 19    Step 59    Step 113    Step 169    Step 190

Figure 5: Real-time MDL hypothesis evolution for actual Concept 11 data. As the size of the data set grows beyond 150, there is oscillation between the one-rectangle (2×4) hypothesis shown in Step 169 and the two-rectangle (1×3) hypothesis shown in Step 190.

# 8 Conclusions

As discussed above, MDL bears a tight relationship with Bayesian inference, and hence serves as a reasonable basis for a theory of learning. The data presented above suggest that human learners are indeed guided by something very much like Rissanen's principle when classifying objects. While it is premature to conclude that humans construct anything precisely corresponding to the two-part code of Equation 4, it seems likely that they employ some closely related complexity-minimization principle—and an associated "cognitive code" still to be discovered. This finding is consistent with many earlier observations of minimum principles guiding human inference, especially in perception (e.g., the Gestalt principle of *Prägnanz*). Moreover, our findings suggest a principled approach to predicting the subjective difficulty of concepts defined over continuous features. As we had previously found with Boolean concepts, subjective difficulty correlates with intrinsic complexity: *That which is incompressible is, in turn, incomprehensible.* The MDL approach is an elegant framework in which to make this observation rigorous and concrete, and one which apparently accords well with human performance.

**Acknowledgments**

This research was supported by NSF SBR-9875175.

**References**

[1] Nosofsky, R. M., "Exemplar-based accounts of relations between classification, recognition, and typicality," *Journal of Experimental Psychology: Learning, Memory, and Cognition,* Vol. 14, No. 4, 1988, pp. 700–708.

[2] Ashby, F. G. and Alfonso-Reese, L. A., "Categorization as probability density estimation," *Journal of Mathematical Psychology*, Vol. 39, 1995, pp. 216–233.

[3] Anderson, J. R., "The adaptive nature of human categorization," *Psychological Review*, Vol. 98, No. 3, 1991, pp. 409–429.

[4] Tenenbaum, J. B., "Bayesian modeling of human concept learning," *Advances in Neural Information Processing Systems*, edited by M. S. Kearns, S. A. Solla, and D. A. Cohn, Vol. 11, MIT Press, Cambridge, MA, 1999.

[5] Nosofsky, R. M., "Optimal performance and exemplar models of classification," *Rational Models of Cognition*, edited by M. Oaksford and N. Chater, chap. 11, Oxford University Press, Oxford, 1998, pp. 218–247.

[6] Nosofsky, R. M., "Further tests of an exemplar-similarity approach to relating identification and categorization," *Perception and Psychophysics*, Vol. 45, 1989, pp. 279–290.

[7] Feldman, J., "The structure of perceptual categories," *Journal of Mathematical Psychology*, Vol. 41, No. 2, 1997, pp. 145–170.

[8] Shepard, R. N., Hovland, C. I., and Jenkins, H. M., "Learning and memorization of classifications," *Psychological Monographs: General and Applied*, Vol. 75, No. 13, 1961, pp. 1–42.

[9] Feldman, J., "Minimization of Boolean complexity in human concept learning," *Nature*, Vol. 407, 2000, pp. 630–632.

[10] Rissanen, J., "Modeling by shortest data description," *Automatica*, Vol. 14, 1978, pp. 465–471.

[11] Li, M. and Vitányi, P., *An Introduction to Kolmogorov Complexity and Its Applications*, Springer, New York, 2nd ed., 1997.

[12] Pothos, E. M. and Chater, N., "Categorization by simplicity: A minimum description length approach to unsupervised clustering," *Similarity and Categorization*, edited by U. Hahn and M. Ramscar, chap. 4, Oxford University Press, Oxford, 2001, pp. 51–72.

[13] Myung, I. J., "Maximum entropy interpretation of decision bound and context models of categorization," *Journal of Mathematical Psychology*, Vol. 38, 1994, pp. 335–365.

[14] Wickens, T. D., *Elementary Signal Detection Theory*, Oxford University Press, Oxford, 2002.

## Footnotes

[1] Equivalently, a model class. The particular choice of language (model class) is obviously an important determinant of the ultimate hypothesis description length. We mention that the MDL analysis in this paper might be replaced by another theoretical approach, such as a Bayesian framework, although we have not pursued this possibility. We adopt the MDL formulation partly because its emphasis on *representation* (i.e., description) seems apt for a study of complexity.

[2] The class "$m \times n$" contains all rectangles of dimension $m \times n$ and $n \times m$.

[3] We use the noninteger value $-\log P(x)$ rather than the integer $\lceil -\log P(x) \rceil$. Logs are base-2.

[4]Note that the MDL hypothesis is *not* in general the most compact hypothesis, i.e., the hypothesis for which $DL(H)$ is a minimum. Rather, the MDL hypothesis is the one for which the sum $DL(D \mid H) + DL(H)$ is minimum.

[5] $d'$ (*discriminability*) gives a measure of subjects' intrinsic capacity to discriminate categories, i.e., one that is independent of their criterion for responding "positive" [14].
